# Non-Local Manifold Tangent Learning

**Yoshua Bengio and Martin Monperrus**
Dept. IRO, Université de Montréal
P.O. Box 6128, Downtown Branch, Montreal, H3C 3J7, Qc, Canada
{bengioy,monperrm}@iro.umontreal.ca

## Abstract

We claim and present arguments to the effect that a large class of manifold learning algorithms that are essentially local and can be framed as kernel learning algorithms will suffer from the curse of dimensionality, at the dimension of the true underlying manifold. This observation suggests to explore non-local manifold learning algorithms which attempt to discover shared structure in the tangent planes at different positions. A criterion for such an algorithm is proposed and experiments estimating a tangent plane prediction function are presented, showing its advantages with respect to local manifold learning algorithms: it is able to generalize very far from training data (on learning handwritten character image rotations), where a local non-parametric method fails.

## 1 Introduction

A central issue of generalization is how information from the training examples can be used to make predictions about new examples, and without strong prior assumptions, i.e. in non-parametric models, this may be fundamentally difficult as illustrated by the curse of dimensionality. There has been in recent years a lot of work on unsupervised learning based on characterizing a possibly non-linear manifold near which the data would lie, such as Locally Linear Embedding (LLE) (Roweis and Saul, 2000), Isomap (Tenenbaum, de Silva and Langford, 2000), kernel Principal Components Analysis (PCA) (Schölkopf, Smola and Müller, 1998), Laplacian Eigenmaps (Belkin and Niyogi, 2003), and Manifold Charting (Brand, 2003). These are all essentially non-parametric methods that can be shown to be kernel methods with an adaptive kernel (Bengio et al., 2004), and which represent the manifold on the basis of local neighborhood relations, very often constructed using the nearest neighbors graph (the graph with one vertex per observed example, and arcs between near neighbors). The above methods characterize the manifold through an embedding which associates each training example (an input object) with a low-dimensional coordinate vector (the coordinates on the manifold). Other closely related methods characterize the manifold as well as "noise" around it. Most of these methods consider the density as a mixture of flattened Gaussians, e.g. mixtures of factor analyzers (Ghahramani and Hinton, 1996), Manifold Parzen windows (Vincent and Bengio, 2003), and other local PCA models such as mixtures of probabilistic PCA (Tipping and Bishop, 1999). This is not an exhaustive list, and recent work also combines modeling through a mixture density and dimensionality reduction (Teh and Roweis, 2003; Brand, 2003).

In this paper we claim that there is a fundamental weakness with such kernel methods, due to the **locality of learning**: we show that the local tangent plane of the manifold at a point $x$ is defined based mostly on the near neighbors of $x$ according to some possibly data-

dependent kernel $K_D$. As a consequence, it is difficult with such methods to generalize to new combinations of values $x$ that are "far" from the training examples $x_i$, where being "far" is a notion that should be understood in the context of several factors: the amount of noise around the manifold (the examples do not lie exactly on the manifold), the curvature of the manifold, and the dimensionality of the manifold. For example, if the manifold curves quickly around $x$, neighbors need to be closer for a locally linear approximation to be meaningful, which means that more data are needed. Dimensionality of the manifold compounds that problem because the amount of data thus needed will grow exponentially with it. Saying that $y$ is "far" from $x$ means that $y$ is not well represented by its projection on the tangent plane at $x$.

In this paper we explore one way to address that problem, based on estimating the tangent planes of the manifolds as a function of $x$, with parameters that can be estimated not only from the data around $x$ but from the whole dataset. Note that there can be more than one manifold (e.g. in vision, one may imagine a different manifold for each "class" of object), but the structure of these manifolds may be related, something that many previous manifold learning methods did not take advantage of. We present experiments on a variety of tasks illustrating the weaknesses of the local manifold learning algorithms enumerated above. The most striking result is that the model is able to generalize a notion of rotation learned on one kind of image (digits) to a very different kind (alphabet characters), i.e. very far from the training data.

## 2   Local Manifold Learning

By "local manifold learning", we mean a method that derives information about the local structure of the manifold (i.e. implicitly its tangent directions) at $x$ based mostly on the training examples "around" x. There is a large group of manifold learning methods (as well as the spectral clustering methods) that share several characteristics, and can be seen as data-dependent kernel PCA (Bengio et al., 2004). These include LLE (Roweis and Saul, 2000), Isomap (Tenenbaum, de Silva and Langford, 2000), kernel PCA (Schölkopf, Smola and Müller, 1998) and Laplacian Eigenmaps (Belkin and Niyogi, 2003). They first build a data-dependent Gram matrix $M$ with $n \times n$ entries $K_D(x_i, x_j)$ where $D = \{x_1, \ldots, x_n\}$ is the training set and $K_D$ is a data-dependent kernel, and compute the eigenvector-eigenvalue pairs $\{(v_k, \lambda_k)\}$ of $M$. The embedding of the training set is obtained directly from the principal eigenvectors $v_k$ of $M$ (the $i$-th element of $v_k$ gives the $k$-th coordinate of $x_i$'s embedding, possibly scaled by $\sqrt{\frac{\lambda_k}{n}}$, i.e. $e_k(x_i) = v_{ik}$) and the embedding for a new example can be estimated using the Nyström formula (Bengio et al., 2004): $e_k(x) = \frac{1}{\lambda_k} \sum_{i=1}^{n} v_{ki} K_D(x, x_i)$ for the $k$-th coordinate of $x$, where $\lambda_k$ is the $k$-th eigenvalue of $M$ (the optional scaling by $\sqrt{\frac{\lambda_k}{n}}$ would also apply). The above equation says that the embedding for a new example $x$ is a local interpolation of the manifold coordinates of its neighbors $x_i$, with interpolating weights given by $\frac{K_D(x, x_i)}{\lambda_k}$. To see more clearly how the tangent plane may depend only on the neighbors of $x$, consider the relation between the tangent plane and the embedding function: **the tangent plane at $x$ is simply the subspace spanned by the vectors $\frac{\partial e_k(x)}{\partial x}$.** In the case of very "local" kernels like that of LLE, spectral clustering with Gaussian kernel, Laplacian Eigenmaps or kernel PCA with Gaussian kernel, that derivative only depends significantly on the near neighbors of $x$. Consider for example kernel PCA with a Gaussian kernel: then $\frac{\partial e_k(x)}{\partial x}$ can be closely approximated by a linear combination of the difference vectors $(x - x_j)$ for $x_j$ near $x$. The weights of that combination may depend on the whole data set, but if the ambient space has many more dimensions then the number of such "near" neighbors of $x$, this is a very strong locally determined constraint on the shape of the manifold. The case of Isomap is less obvious but we show below that it is also local. Let $\mathcal{D}(a, b)$ denote the graph geodesic distance going only through $a$, $b$ and points from the training set. As

shown in (Bengio et al., 2004), the corresponding data-dependent kernel can be defined as $K_D(x, x_i) = -\frac{1}{2}(\mathcal{D}(x, x_i)^2 - \frac{1}{n}\sum_j \mathcal{D}(x, x_j)^2 - \bar{\mathcal{D}}_i + \bar{\mathcal{D}})$ where $\bar{\mathcal{D}}_i = \frac{1}{n}\sum_j \mathcal{D}(x_i, x_j)^2$ and $\bar{\mathcal{D}} = \frac{1}{n}\sum_j \bar{\mathcal{D}}_j$. Let $\mathcal{N}(x, x_i)$ denote the index $j$ of the training set example $x_j$ that is a neighbor of $x$ that minimizes $||x - x_j|| + \mathcal{D}(x_j, x_i)$. Then

$$\frac{\partial e_k(x)}{\partial x} = \frac{1}{\lambda_k}\sum_i v_{ki}\left(\frac{1}{n}\sum_j \mathcal{D}(x, x_j)\frac{(x - x_{\mathcal{N}(x,x_j)})}{||x - x_{\mathcal{N}(x,x_j)}||} - \mathcal{D}(x, x_i)\frac{(x - x_{\mathcal{N}(x,x_i)})}{||x - x_{\mathcal{N}(x,x_i)}||}\right)$$
(1)

which is a linear combination of vectors $(x - x_k)$, where $x_k$ is a neighbor of $x$. *This clearly shows that the tangent plane at $x$ associated with Isomap is also included in the subspace spanned by the vectors $(x - x_k)$ where $x_k$ is a neighbor of $x$.*

There are also a variety of local manifold learning algorithms which can be classified as "mixtures of pancakes" (Ghahramani and Hinton, 1996; Tipping and Bishop, 1999; Vincent and Bengio, 2003; Teh and Roweis, 2003; Brand, 2003). These are generally mixtures of Gaussians with a particular covariance structure. When the covariance matrix is approximated using its principal eigenvectors, this leads to "local PCA" types of methods. For these methods the local tangent directions directly correspond to the principal eigenvectors of the local covariance matrices. Learning is also local since it is mostly the examples around the Gaussian center that determine its covariance structure. The problem is not so much due to the form of the density as a mixture of Gaussians. The problem is that the local parameters (e.g. local principal directions) are estimated mostly based on local data. There is usually a non-local interaction between the different Gaussians, but its role is mainly of global coordination, e.g. where to set the Gaussian centers to allocate them properly where there is data, and optionally how to orient the principal directions so as to obtain a globally coherent coordinate system for embedding the data.

### 2.1 Where Local Manifold Learning Would Fail

It is easy to imagine at least four failure causes for local manifold learning methods, and combining them will create even greater problems:

• **Noise around the manifold**: data are not exactly lying on the manifold. In the case of non-linear manifolds, the presence of noise means that more data around each pancake region will be needed to properly estimate the tangent directions of the manifold in that region.

• **High curvature of the manifold**. Local manifold learning methods basically approximate the manifold by the union of many locally linear patches. For this to work, there must be at least $d$ close enough examples in each patch (more with noise). With a high curvature manifold, more – smaller – patches will be needed, and the number of required patches will grow exponentially with the dimensionality of the manifold. Consider for example the manifold of translations of a high-contrast image.The tangent direction corresponds to the change in image due a small translation, i.e. it is non-zero only at edges in the image. After a one-pixel translation, the edges have moved by one pixel, and may not overlap much with the edges of the original image if it had high contrast. This is indeed a very high curvature manifold.

• **High intrinsic dimension of the manifold**. We have already seen that high manifold dimensionality $d$ is hurtful because $O(d)$ examples are required in each patch and $O(r^d)$ patches (for some $r$ depending on curvature and noise) are necessary to span the manifold. In the translation example, if the image resolution is increased, then many more training images will be needed to capture the curvature around the translation manifold with locally linear patches. Yet the physical phenomenon responsible for translation is expressed by a simple equation, which does not get more complicated with increasing resolution.

• **Presence of many manifolds with little data per manifold**. In many real-world contexts there is not just one global manifold but a large number of manifolds which however

share something about their structure. A simple example is the manifold of transformations (view-point, position, lighting,...) of 3D objects in 2D images. There is one manifold per object instance (corresponding to the successive application of small amounts of all of these transformations). If there are only a few examples for each such class then it is almost impossible to learn the manifold structures using only local manifold learning. However, if the manifold structures are generated by a common underlying phenomenon then a non-local manifold learning method could potentially learn all of these manifolds and even generalize to manifolds for which a single instance is observed, as demonstrated in the experiments below.

## 3  Non-Local Manifold Tangent Learning

Here we choose to characterize the manifolds in the data distribution through a matrix-valued function $F(x)$ that predicts at $x \in \mathbf{R}^n$ a basis for the tangent plane of the manifold near $x$, hence $F(x) \in \mathbf{R}^{d \times n}$ for a $d$-dimensional manifold. Basically, $F(x)$ specifies "where" (in which directions) one expects to find near neighbors of $x$.

We are going to consider a simple supervised learning setting to train this function. As with Isomap, we consider that the vectors $(x - x_i)$ with $x_i$ a near neighbor of $x$ span a noisy estimate of the manifold tangent space. We propose to use them to define a "target" for training $F(x)$. In our experiments we simply collected the $k$ nearest neighbors of each example $x$, but better selection criteria might be devised. Points on the predicted tangent subspace can be written $F'(x)w$ with $w \in \mathbf{R}^d$ being local coordinates in the basis specified by $F(x)$. Several criteria are possible to match the neighbors differences with the subspace defined by $F(x)$. One that yields to simple analytic calculations is simply to minimize the distance between the $x - x_j$ vectors and their projection on the subspace defined by $F(x)$. The low-dimensional local coordinate vector $w_{tj} \in \mathbf{R}^d$ that matches neighbor $x_j$ of example $x_t$ is thus an extra free parameter that has to be optimized, but is obtained analytically. The overall training criterion involves a double optimization over function $F$ and local coordinates $w_{tj}$ of what we call the **relative projection error**:

$$\min_{F, \{w_{tj}\}} \sum_t \sum_{j \in \mathcal{N}(x_t)} \frac{||F'(x_t)w_{tj} - (x_t - x_j)||^2}{||x_t - x_j||^2} \tag{2}$$

where $\mathcal{N}(x)$ denotes the selected set of near neighbors of $x$. The normalization by $||x_t - x_j||^2$ is to avoid giving more weight to the neighbors that are further away. The above ratio amounts to minimizing the square of the sinus of the projection angle. To perform the above minimization, we can do coordinate descent (which guarantees convergence to a minimum), i.e. alternate changes in $F$ and changes in $w$'s which at each step go down the total criterion. Since the minimization over the $w$'s can be done separately for each example $x_t$ and neighbor $x_j$, it is equivalent to minimize

$$\frac{||F'(x_t)w_{tj} - (x_t - x_j)||^2}{||x_t - x_j||^2} \tag{3}$$

over vector $w_{tj}$ for each such pair (done analytically) and compute the gradient of the above over $F$ (or its parameters) to move $F$ slightly (we used stochastic gradient on the parameters of $F$). The solution for $w_{tj}$ is obtained by solving the linear system

$$F(x_t)F'(x_t)w_{tj} = F(x_t)\frac{(x_t - x_j)}{||x_t - x_j||^2}. \tag{4}$$

In our implementation this is done robustly through a singular value decomposition $F'(x_t) = USV'$ and $w_{tj} = B(x_t - x_j)$ where $B$ can be precomputed for all the neighbors of $x_t$: $B = (\sum_{k=1}^d 1_{S_k > \epsilon} V_{.k} V'_{.k} / S_k^2) F(x_t)$. The gradient of the criterion with respect to

the $i$-th row of $F(x_t)$, holding $w_{tj}$ fixed, is simply

$$2 \sum_j \frac{w_{tji}}{||x_t - x_j||} (F'(x_t)w - (x_t - x_j))$$  (5)

where $w_{tji}$ is the $i$-th element of $w_{tj}$. In practice, it is not necessary to store more than one $w_{tj}$ vector at a time. In the experiments, $F(\cdot)$ is parameterized as a an ordinary one hidden layer neural network with $n$ inputs and $d \times n$ outputs. It is trained by stochastic gradient descent, one example $x_t$ at a time.

Although the above algorithm provides a characterization of the manifold, it does not directly provide an embedding nor a density function. However, once the tangent plane function is trained, there are ways to use it to obtain all of the above. The simplest method is to apply existing algorithms that provide both an embedding and a density function based on a Gaussian mixture with pancake-like covariances. For example one could use (Teh and Roweis, 2003) or (Brand, 2003), the local covariance matrix around $x$ being constructed from $F'(x)diag(\sigma^2(x))F(x)$, where $\sigma_i^2(x)$ should estimate $Var(w_i)$ around $x$.

### 3.1 Previous Work on Non-Local Manifold Learning

The non-local manifold learning algorithm presented here (find $F(\cdot)$ which minimizes the criterion in eq. 2) is similar to the one proposed in (Rao and Ruderman, 1999) to estimate the generator matrix of a Lie group. That group defines a one-dimensional manifold generated by following the orbit $x(t) = e^{Gt}x(0)$, where $G$ is an $n \times n$ matrix and $t$ is a scalar manifold coordinate. A multi-dimensional manifold can be obtained by replacing $Gt$ above by a linear combination of multiple generating matrices. In (Rao and Ruderman, 1999) the matrix exponential is approximated to first order by $(I + Gt)$, and the authors estimate $G$ for a simple signal undergoing translations, using as a criterion the minimization of $\sum_{x,\tilde{x}} \min_t ||(I + Gt)x - \tilde{x}||^2$, where $\tilde{x}$ is a neighbor of $x$ in the data. Note that in this model the tangent plane is a linear function of $x$, i.e. $F_1(x) = Gx$. By minimizing the above across many pairs of examples, a good estimate of $G$ for the artificial data was recovered by (Rao and Ruderman, 1999). Our proposal extends this approach to multiple dimensions and non-linear relations between $x$ and the tangent planes. Note also the earlier work on Tangent Distance (Simard, LeCun and Denker, 1993), in which the tangent planes are not learned but used to build a nearest neighbor classifier that is based on the distance between the tangent subspaces around two examples to be compared. The main advantage of the approach proposed here over local manifold learning is that the parameters of the tangent plane predictor can be estimated using data from very different regions of space, thus in principle allowing to be less sensitive to all four of the problems described in 2.1, thanks to sharing of information across these different regions.

## 4 Experimental Results

The objective of the experiments is to validate the proposed algorithm: does it estimate well the true tangent planes? does it learn better than a local manifold learning algorithm?

**Error Measurement** In addition to visualizing the results for the low-dimensional data, we measure performance by considering how well the algorithm learns the local tangent distance, as measured by the normalized projection error of nearest neighbors (eq. 3). We compare the errors of four algorithms, always on test data not used to estimate the tangent plane: (a) **true analytic** (using the true manifold's tangent plane at $x$ computed analytically), (b) **tangent learning** (using the neural-network tangent plane predictor $F(x)$, trained using the $k \geq d$ nearest neighbors in the training set of each training set example), (c) **Isomap** (using the tangent plane defined on Eq. 1), (d) **Local PCA** (using the $d$ principal components of the $k$ nearest neighbors of $x$ in the training set).

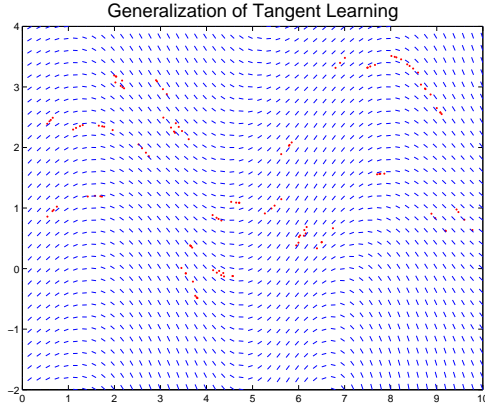

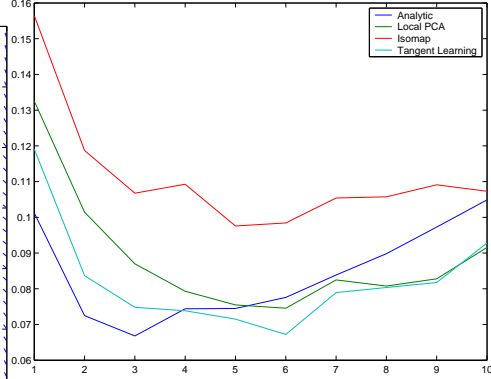

Figure 1: *Task 1 2-D data with 1-D sinusoidal manifolds: the method indeed captures the tangent planes. The small segments are the estimated tangent planes. Red points are training examples.*

Figure 2: *Task 2 relative projection error for k-th nearest neighbor, w.r.t. k, for compared methods (from lowest to highest at k=1: analytic, tangent learning, local PCA, Isomap). Note U-shape due to opposing effects of curvature and noise.*

**Task 1** We first consider a low-dimensional but multi-manifold problem. The data $\{x_i\}$ are in 2 dimensions and coming from a set of 40 1-dimensional manifolds. Each manifold is composed of 4 near points obtained from a randomly based sinus, i.e $\forall i \in 1..4$, $x_i = (a + t_i, sin(a + t_i) + b)$, where $a$, $b$, and $t_i$ are randomly chosen. Four neighbors were used for training both the Tangent Learning algorithm and the benchmark local non-parametric estimator (local PCA of the 4 neighbors). Figure 1 shows the training set and the tangent planes recovered, both on the training examples and generalizing away from the data. The neural network has 10 hidden units (chosen arbitrarily). This problem is particularly difficult for local manifold learning, which does very poorly here: the out-of-sample relative prediction error are respectively 0.09 for the **true analytic** plane, 0.25 for **non-local tangent learning**, and 0.81 for **local PCA**.

**Task 2** This is a higher dimensional manifold learning problem, with 41 dimensions. The data are generated by sampling Gaussian curves. Each curve is of the form $x(i) = e^{t_1 - (-2+i/10)^2/t_2}$ with $i \in \{0, 1, \dots, 40\}$. Note that the tangent vectors are not linear in $x$. The manifold coordinates are $t_1$ and $t_2$, sampled uniformly, respectively from $(-1, 1)$ and $(.1, 3.1)$. Normal noise (standard deviation = 0.001) is added to each point. 100 example curves were generated for training and 200 for testing. The neural network has 100 hidden units. Figure 2 shows the relative projection error for the four methods on this task, for the $k$-th nearest neighbor, for increasing values of $k$. First, the error decreases because of the effect of noise (near noisy neighbors may form a high angle with the tangent plane). Then, it increases because of the curvature of manifold (further away neighbors form a larger angle).

**Task 3** This is a high-dimensional multi-manifold task, involving **digit images** to which we have applied slight rotations, in such a way as to have the knowledge of the analytic formulation of the manifolds. There is one rotation manifold for each instance of digit from the database, but only two examples for each manifold: one real image from the MNIST dataset and one slightly rotated image. $1000 \times 2$ examples are used for training and $1000 \times 2$ for testing. In this context we use $k = 1$ nearest neighbor only and manifold dimension is 1. The average relative projection error for the nearest neighbor is 0.27 for the **analytic** tangent plane, 0.43 with **tangent learning** (100 hidden units), and 1.5 with **Local PCA**. Here the neural network would probably overfit if trained too much (here only 100 epochs).

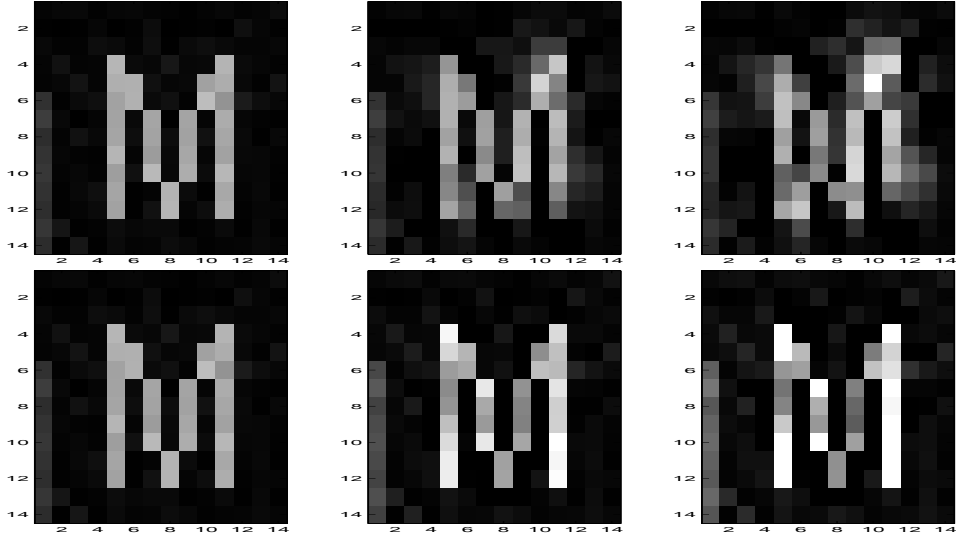

Figure 3: Left column: original image. Middle: applying a small amount of the predicted rotation. Right: applying a larger amount of the predicted rotation. Top: using the estimated tangent plane predictor. Bottom: using local PCA, which is clearly much worse.

An even more interesting experiment consists in ***applying the above trained predictor on novel images that come from a very different distribution but one that shares the same manifold structure***: it was applied to images of other characters that are not digits. We have used the predicted tangent planes to follow the manifold by small steps (this is very easy to do in the case of a one-dimensional manifold). Figure 3 shows for example on a letter 'M' image the effect of a few such steps and a larger number of steps, both for the neural network predictor and for the local PCA predictor.

This example illustrates the crucial point that non-local tangent plane learning is able to generalize to truly novel cases, where local manifold learning fails.

In all the experiments we found that all the randomly initialized neural networks converged to similarly good solutions. The number of hidden units was not optimized, although preliminary experimentation showed phenomena of over-fitting and under-fitting due to too small or too large number hidden units was possible.

## 5  Conclusion

The central claim of this paper is that there are fundamental problems with non-parametric local approaches to manifold learning, essentially due to the curse of dimensionality (at the dimensionality of the manifold), but worsened by manifold curvature, noise, and the presence of several disjoint manifolds. To address these problems, we propose that learning algorithms should be designed in such a way that they can share information, coming from different regions of space, about the structure of the manifold. In this spirit we have proposed a simple learning algorithm based on predicting the tangent plane at $x$ with a function $F(x)$ whose parameters are estimated based on the whole data set. Note that the same fundamental problems are present with non-parametric approaches to semi-supervised learning (e.g. as in (Szummer and Jaakkola, 2002; Chapelle, Weston and Scholkopf, 2003; Belkin and Niyogi, 2003; Zhu, Ghahramani and Lafferty, 2003)), which rely on proper estimation of the manifold in order to propagate label information.

Future work should investigate how to better handle the curvature problem, e.g. by follow-

ing the manifold (using the local tangent estimates), to estimate a manifold-following path between pairs of neighboring examples. The algorithm can also be extended in a straightforward way to obtain a Gaussian mixture or a mixture of factor analyzers (with the factors or the principal eigenvectors of the Gaussian centered at $x$ obtained from $F(x)$). This view can also provide an alternative criterion to optimize $F(x)$ (the local log-likelihood of such a Gaussian). This criterion also tells us how to estimate the missing information (the variances along the eigenvector directions). Since we can estimate $F(x)$ everywhere, a more ambitious view would consider the density as a "continuous" mixture of Gaussians (with an infinitesimal component located everywhere in space).

## Acknowledgments

The authors would like to thank the following funding organizations for support: NSERC, MITACS, IRIS, and the Canada Research Chairs.

# References

Belkin, M. and Niyogi, P. (2003). Using manifold structure for partially labeled classification. In Becker, S., Thrun, S., and Obermayer, K., editors, *Advances in Neural Information Processing Systems 15*, Cambridge, MA. MIT Press.

Bengio, Y., Delalleau, O., Le Roux, N., Paiement, J.-F., Vincent, P., and Ouimet, M. (2004). Learning eigenfunctions links spectral embedding and kernel PCA. *Neural Computation*, 16(10):2197–2219.

Brand, M. (2003). Charting a manifold. In Becker, S., Thrun, S., and Obermayer, K., editors, *Advances in Neural Information Processing Systems 15*. MIT Press.

Chapelle, O., Weston, J., and Scholkopf, B. (2003). Cluster kernels for semi-supervised learning. In Becker, S., Thrun, S., and Obermayer, K., editors, *Advances in Neural Information Processing Systems 15*, Cambridge, MA. MIT Press.

Ghahramani, Z. and Hinton, G. (1996). The EM algorithm for mixtures of factor analyzers. Technical Report CRG-TR-96-1, Dpt. of Comp. Sci., Univ. of Toronto.

Rao, R. and Ruderman, D. (1999). Learning lie groups for invariant visual perception. In Kearns, M., Solla, S., and Cohn, D., editors, *Advances in Neural Information Processing Systems 11*, pages 810–816. MIT Press, Cambridge, MA.

Roweis, S. and Saul, L. (2000). Nonlinear dimensionality reduction by locally linear embedding. *Science*, 290(5500):2323–2326.

Schölkopf, B., Smola, A., and Müller, K.-R. (1998). Nonlinear component analysis as a kernel eigenvalue problem. *Neural Computation*, 10:1299–1319.

Simard, P., LeCun, Y., and Denker, J. (1993). Efficient pattern recognition using a new transformation distance. In Giles, C., Hanson, S., and Cowan, J., editors, *Advances in Neural Information Processing Systems 5*, pages 50–58, Denver, CO. Morgan Kaufmann, San Mateo.

Szummer, M. and Jaakkola, T. (2002). Partially labeled classification with markov random walks. In Dieterich, T., Becker, S., and Ghahramani, Z., editors, *Advances in Neural Information Processing Systems 14*, Cambridge, MA. MIT Press.

Teh, Y. W. and Roweis, S. (2003). Automatic alignment of local representations. In Becker, S., Thrun, S., and Obermayer, K., editors, *Advances in Neural Information Processing Systems 15*. MIT Press.

Tenenbaum, J., de Silva, V., and Langford, J. (2000). A global geometric framework for nonlinear dimensionality reduction. *Science*, 290(5500):2319–2323.

Tipping, M. and Bishop, C. (1999). Mixtures of probabilistic principal component analysers. *Neural Computation*, 11(2):443–482.

Vincent, P. and Bengio, Y. (2003). Manifold parzen windows. In Becker, S., Thrun, S., and Obermayer, K., editors, *Advances in Neural Information Processing Systems 15*, Cambridge, MA. MIT Press.

Zhu, X., Ghahramani, Z., and Lafferty, J. (2003). Semi-supervised learning using gaussian fields and harmonic functions. In *ICML'2003*.
